# Critical Lines in Symmetry of Mixture Models and its Application to Component Splitting

**Kenji Fukumizu**
Institute of Statistical
Mathematics
Tokyo 106-8569 Japan
*fukumizu@ism.ac.jp*

**Shotaro Akaho**
AIST
Tsukuba 305-8568 Japan
*s.akaho@aist.go.jp*

**Shun-ichi Amari**
RIKEN
Wako 351-0198 Japan
*amari@brain.riken.go.jp*

## Abstract

We show the existence of critical points as lines for the likelihood function of mixture-type models. They are given by embedding of a critical point for models with less components. A sufficient condition that the critical line gives local maxima or saddle points is also derived. Based on this fact, a component-split method is proposed for a mixture of Gaussian components, and its effectiveness is verified through experiments.

## 1  Introduction

The likelihood function of a mixture model often has a complex shape so that calculation of an estimator can be difficult, whether the maximum likelihood or Bayesian approach is used. In the maximum likelihood estimation, convergence of the EM algorithm to the global maximum is not guaranteed, while it is a standard method. Investigation of the likelihood function for mixture models is important to develop effective methods for learning.

This paper discusses the critical points of the likelihood function for mixture-type models by analyzing their hierarchical symmetric structure. As generalization of [1], we show that, given a critical point of the likelihood for the model with $(H-1)$ components, duplication of any of the components gives critical points as lines for the model with $H$ components. We call them *critical lines* of mixture models. We derive also a sufficient condition that the critical lines give maxima or saddle points of the larger model, and show that given a maximum of the likelihood for a mixture of Gaussian components, an appropriate split of any component always gives an ascending direction of the likelihood. Based on this theory, we propose a stable method of splitting a component, which works effectively with the EM optimization for avoiding the dependency on the initial condition and improving the optimization. The usefulness of the algorithm is verified through experiments.

## 2  Hierarchical Symmetry and Critical Lines of Mixture Models

### 2.1  Symmetry of Mixture models

Suppose $f_H(x \mid \boldsymbol{\theta}^{(H)})$ is a mixture model with $H$ components, defined by

$$f_H(x \mid \boldsymbol{\theta}^{(H)}) = \sum_{j=1}^{H} c_j \, p(x \mid \beta_j), \qquad c_j = \alpha_j/(\alpha_1 + \cdots + \alpha_H), \qquad (1)$$

where $p(x \mid \beta)$ is a probability density function with a parameter $\beta$. We write, for simplicity, $\boldsymbol{\alpha}^{(H)} = (\alpha_1, \ldots, \alpha_H)$, $\boldsymbol{\beta}^{(H)} = (\beta_1, \ldots, \beta_H)$, and $\boldsymbol{\theta}^{(H)} = (\boldsymbol{\alpha}^{(H)}; \boldsymbol{\beta}^{(H)})$.

The key of our discussion is the following two symmetric properties, which are satisfied by mixture models;

**(S-1)** $f_H(x \mid \boldsymbol{\alpha}^{(H)}; \boldsymbol{\beta}^{(H-2)}, \beta_{H-1}, \beta_{H-1}) = f_{H-1}(x \mid \boldsymbol{\alpha}^{(H-2)}, \alpha_{H-1} + \alpha_H; \boldsymbol{\beta}^{(H-1)})$.

**(S-2)** There exists a function $A(\boldsymbol{\alpha})$ such that for $j = H - 1$ and $H$,
$$\frac{\partial f_H}{\partial \beta_j}(x \mid \boldsymbol{\alpha}^{(H)}; \boldsymbol{\beta}^{(H-2)}, \beta_{H-1}, \beta_{H-1}) = \frac{\alpha_j}{A(\boldsymbol{\alpha})} \frac{\partial f_{H-1}}{\partial \beta_{H-1}}(x \mid \boldsymbol{\alpha}^{(H-2)}, \alpha_{H-1} + \alpha_H; \boldsymbol{\beta}^{(H-1)}).$$

In mixture models, the function $A(\alpha)$ is simply given by $A(\alpha) = \alpha_1 + \cdots + \alpha_H$.

Hereafter, we discuss in general a model with the assumptions (S-1) and (S-2). The results in Sections 2.1 and 2.2 depend only on these assumptions [1]. While in mixture models similar conditions are satisfied with any choices of two components, we describe only the case of $H - 1$ and $H$ just for simplicity. We write $\Theta_H$ for the space of the parameter $\boldsymbol{\theta}^{(H)}$.

Another example which satisfies (S-1) and (S-2) is Latent Dirichlet Allocation (LDA, [2]), which models data of a group structure (e.g. document as a set of words). For $x = (x_1, \ldots, x_M)$, LDA with $H$ components is defined by

$$f_H(x \mid \boldsymbol{\theta}^{(H)}) = \int_{\Delta_{H-1}} \mathcal{D}_H(\boldsymbol{u}^{(H)} \mid \boldsymbol{\alpha}^{(H)}) \prod_{\nu=1}^{M} \left( \sum_{j=1}^{H} u_j p(x_\nu \mid \beta_j) \right) d\boldsymbol{u}^{(H)}, \qquad (2)$$

where $\mathcal{D}_H(\boldsymbol{u}^{(H)} \mid \boldsymbol{\alpha}^{(H)}) = \frac{\Gamma(\sum_j \alpha_j)}{\prod_j \Gamma(\alpha_j)} \prod_{j=1}^{H} u_j^{\alpha_j - 1}$ is the Dirichlet distribution over the $(H-1)$-dimensional simplex $\Delta_{H-1}$. It is easy to see (S-1) and (S-2) hold for LDA by using Lemma 6 in Appendix. LDA includes mixture models eq.(1) as the special case of $M = 1$.

It is straightforward from (S-1) that, given a parameter $\boldsymbol{\theta}^{(H-1)} = (\boldsymbol{\gamma}^{(H-1)}; \boldsymbol{\eta}^{(H-1)})$ of the model with $(H-1)$ components and a scalar $\lambda$, the parameter $\boldsymbol{\theta}_\lambda \in \Theta_H$ defined by

$$\begin{aligned} \alpha_j = \gamma_j, &\qquad\qquad \beta_j = \eta_j \qquad (1 \le j \le H - 2) \\ \alpha_{H-1} = \lambda \gamma_{H-1}, \quad \alpha_H = (1-\lambda)\gamma_{H-1}, &\qquad \beta_{H-1} = \beta_H = \eta_{H-1} \end{aligned} \qquad (3)$$

gives the same function as $f_{H-1}(x \mid \boldsymbol{\theta}^{(H-1)})$. In mixture models/LDA, this corresponds to duplication of the $(H-1)$-th component with partitioning the mixing/Dirichlet parameter in the ratio $\lambda : (1 - \lambda)$. Since $\lambda$ is arbitrary, a point in the smaller model is embedded into the larger model as a line in the parameter space $\Theta_H$. This implies that the parameter to realize $f_{H-1}(x \mid \boldsymbol{\theta}^{(H-1)})$ lacks identifiability in $\Theta_H$. Such singular structure of a model causes various interesting phenomena in estimation, learning, and generalization ([3]).

## 2.2 Critical Lines – Embedding of a Critical Point

Given a sample $\{X^{(1)}, \ldots, X^{(N)}\}$, we define an objective function for learning by

$$L_H(\boldsymbol{\theta}^{(H)}) = \sum_{n=1}^{N} \Omega_n(f_H(X^{(n)} \mid \boldsymbol{\theta}^{(H)})), \qquad (4)$$

where $\Omega_n(f)$ are differentiable functions, which may depend on $n$. The objective of learning is to maximize $L_H$. If $\Omega_n(f) = \log f$ for all $n$, maximization of $L_H(\boldsymbol{\theta}^{(H)})$ is equal to the maximum likelihood estimation.

Suppose $\boldsymbol{\theta}_*^{(H-1)} = (\gamma_1^*, \ldots, \gamma_{H-1}^*; \eta_1^*, \ldots, \eta_{H-1}^*)$ is a critical point of $L_{H-1}(\boldsymbol{\theta}^{(H-1)})$, that is, $\frac{\partial L_{H-1}}{\partial \boldsymbol{\theta}^{(H-1)}}(\boldsymbol{\theta}_*^{(H-1)}) = 0$. Embedding of this point into $\Theta_H$ gives a critical line;

**Theorem 1 (Critical Line).** *Suppose that a model satisfies (S-1) and (S-2). Let $\boldsymbol{\theta}_*^{(H-1)}$ be a critical point of $L_{H-1}$ with $\gamma_{H-1}^* \neq 0$, and $\boldsymbol{\theta}_\lambda$ be a parameter given by eq.(3) for $\boldsymbol{\theta}_*^{(H-1)}$. Then, $\boldsymbol{\theta}_\lambda$ is a critical point of $L_H(\boldsymbol{\theta}^{(H)})$ for all $\lambda$.*

*Proof.* Although this is essentially the same as Theorem 1 in [1], the following proof gives better intuition. Let $(s, t; \zeta, \xi)$ be reparametrization of $(\alpha_{H-1}, \alpha_H; \beta_{H-1}, \beta_H)$, defined by

$$s = \alpha_{H-1} + \alpha_H, \quad t = \alpha_{H-1} - \alpha_H, \quad \beta_{H-1} = \zeta + \alpha_H \xi, \quad \beta_H = \zeta - \alpha_{H-1}\xi. \quad (5)$$

This is a one-to-one correspondence, if $\alpha_{H-1} + \alpha_H \neq 0$. Note that $\xi = 0$ is equivalent to the condition $\beta_{H-1} = \beta_H$. Let $\boldsymbol{\omega} = (\boldsymbol{\alpha}^{(H-2)}, s, t; \boldsymbol{\beta}^{(H-2)}, \zeta, \xi)$ be the new coordinate, $\ell_H(\boldsymbol{\omega})$ be the objective function eq.(4) under this parametrization, and $\boldsymbol{\omega}_\lambda$ be the parameter corresponding to $\boldsymbol{\theta}_\lambda$. Since we have, by definition, $\ell_H(\boldsymbol{\omega}) = L_H(\boldsymbol{\alpha}^{(H-2)}, \frac{s+t}{2}, \frac{s-t}{2}; \boldsymbol{\beta}^{(H-2)}, \zeta + \frac{s-t}{2}\xi, \zeta - \frac{s+t}{2}\xi)$, the condition (S-1) means

$$\ell_H(\boldsymbol{\alpha}^{(H-2)}, s, t; \boldsymbol{\beta}^{(H-2)}, \zeta, 0) = L_{H-1}(\boldsymbol{\alpha}^{(H-2)}, s; \boldsymbol{\beta}^{(H-2)}, \zeta). \quad (6)$$

Then, it is clear that the first derivatives of $\ell_H$ at $\boldsymbol{\omega}_\lambda$ with respect to $\boldsymbol{\alpha}^{(H-2)}, s, \boldsymbol{\beta}^{(H-2)}$, and $\zeta$ are equal to those of $L_{H-1}(\boldsymbol{\theta}^{(H-1)})$ at $\boldsymbol{\theta}_*^{(H-1)}$, and they are zero. The derivative $\partial \ell_H(\boldsymbol{\omega}_\lambda)/\partial t$ vanishes from eq.(6), and $\partial \ell_H(\boldsymbol{\omega}_\lambda)/\partial \xi = 0$ from following Lemma 2. $\square$

**Lemma 2.** *Let $\mathcal{H}$ be a hyperplane given by $\{\boldsymbol{\omega} \mid \xi = 0\}$. Then, for all $\boldsymbol{\omega}_o \in \mathcal{H}$, we have*

$$\frac{\partial f_H}{\partial \xi}(x \mid \boldsymbol{\omega}_o) = 0. \quad (7)$$

*Proof.* Straightforward from the assumption (S-2) and $\frac{\partial}{\partial \xi} = \alpha_H \frac{\partial}{\partial \beta_{H-1}} - \alpha_{H-1} \frac{\partial}{\partial \beta_H}$. $\square$

Given that a maximum of $L_H$ is larger than that of $L_{H-1}$, Theorem 1 implies that the function $L_H$ always has critical points which are not global maximum. Those points lie on lines in the parameter space. Further embedding of the critical lines into larger models gives high-dimensional critical planes in the parameter space. This property is very general, and in LDA and mixture models we do not need any assumptions on $p(x \mid \beta)$. In these models, by the permutation symmetry of components, there are many choices for embedding, which induces many critical lines and planes for $L_H$.

## 2.3 Embedding of a Maximum Point in LDA and Mixture Models

The next question is whether or not the critical lines from a maximum of $L_{H-1}$ gives maxima of $L_H$. The answer requires information on the second derivatives, and depends on models. We show a general result on LDA, and that on mixture models as its corollary.

**Theorem 3.** *Suppose that the model is LDA defined by eq.(2). Let $\boldsymbol{\theta}_*^{(H-1)}$ be an isolated maximum point of $L_{H-1}$, and $\boldsymbol{\theta}_\lambda$ be its embedding given by eq.(3). Define a symmetric matrix $R$ of the size $\dim \beta$ by*

$$R = \sum_{n=1}^N \Omega_n'(f_{H-1}(X^{(n)} \mid \boldsymbol{\theta}_*^{(H-1)}))\Big\{\sum_{\mu=1}^M I_\mu^{(n)} \frac{\partial^2 p(X_\mu^{(n)} \mid \eta_{H-1}^*)}{\partial \beta \partial \beta}$$

$$+ \frac{1}{\sum_{j=1}^{H-1}\gamma_j^*+1}\sum_{\mu=1}^M \sum_{\substack{\tau=1 \\ \tau\neq\mu}}^M J_{\mu,\tau}^{(n)} \frac{\partial p(X_\mu^{(n)} \mid \eta_{H-1}^*)}{\partial \beta} \frac{\partial p(X_\tau^{(n)} \mid \eta_{H-1}^*)}{\partial \beta}\Big\},$$

*where $\Omega'(f)$ denotes the derivative of $\Omega(f)$ w.r.t. $f$, and*

$$I_\mu^{(n)} = \int_{\Delta_{H-2}} \mathcal{D}_{H-1}(\boldsymbol{u} \mid \gamma_1^*, \ldots, \gamma_{H-2}^*, \gamma_{H-1}^* + 1) \prod_{\nu\neq\mu}\Big(\sum_{j=1}^{H-1}u_j p(X_\nu^{(n)} \mid \beta_j)\Big)d\boldsymbol{u}^{(H-1)},$$

$$J_{\mu,\tau}^{(n)} = \int_{\Delta_{H-2}} \mathcal{D}_{H-1}(\boldsymbol{u} \mid \gamma_1^*, \ldots, \gamma_{H-2}^*, \gamma_{H-1}^* + 2) \prod_{\nu\neq\mu,\tau}\Big(\sum_{j=1}^{H-1}u_j p(X_\nu^{(n)} \mid \beta_j)\Big)d\boldsymbol{u}^{(H-1)}.$$

*Then, we have*
*(i) If $R$ is negative definite, the parameter $\boldsymbol{\theta}_\lambda$ is a maximum of $L_H$ for all $\lambda \in (0, 1)$.*
*(ii) If $R$ has a positive eigenvalue, the parameter $\boldsymbol{\theta}_\lambda$ is a saddle point for all $\lambda \in (0, 1)$.*

*Remark:* The conditions on $R$ depend only on the parameter $\theta_*^{(H-1)}$.

*Proof.* We use the parametrization $\boldsymbol{\omega}$ defined by eq.(5). For each $t$, let $\mathcal{H}_t$ be a hyperplane with $t$ fixed, and $\tilde{L}_{H,t}$ be the function $L_H$ restricted on $\mathcal{H}_t$. The hyperplane $\mathcal{H}_t$ is a slice transversal to the critical line, along which $L_H$ has the same value. Therefore, if the Hessian matrix of $\tilde{L}_{H,t}$ on $\mathcal{H}_t$ is negative definite at the intersection $\boldsymbol{\omega}_\lambda$ ($\lambda = (t+1)/2$), the point is a maximum of $L_H$, and if the Hessian has a positive eigenvalue, $\boldsymbol{\omega}_\lambda$ is a saddle point.

Since in $\boldsymbol{\omega}$ coordinate we have $\tilde{L}_{H,t}(\boldsymbol{\alpha}^{(H-1)}, s; \boldsymbol{\beta}^{(H-1)}, \zeta, 0) = L_{H-1}(\boldsymbol{\alpha}^{(H-1)}, s; \boldsymbol{\beta}^{(H-1)}, \zeta)$, the Hessian of $\tilde{L}_{H,t}$ at $\boldsymbol{\omega}_\lambda$ is given by

$$\text{Hess}\tilde{L}_{H,t}(\boldsymbol{\omega}_\lambda) = \begin{pmatrix} \text{Hess}L_{H-1}(\theta_*^{(H-1)}) & O \\ O & \frac{\partial^2 \tilde{L}_{H,t}(\omega_\lambda)}{\partial \xi \partial \xi} \end{pmatrix}. \tag{8}$$

The off-diagonal blocks are zero, because we have $\frac{\partial^2 \tilde{L}_{H,t}(\omega_\lambda)}{\partial \xi \partial \omega_a} = 0$ for $\omega_a \neq \xi$ from Lemma 2. By assumption, $\text{Hess}L_{H-1}(\theta_*^{(H-1)})$ is negative definite. Noting that the terms including $\partial f_H(X^{(n)}; \boldsymbol{\theta}_\lambda)/\partial \xi$ vanish from Lemma 2, it is easy to obtain $\frac{\partial^2 \tilde{L}_{H,t}(\omega_\lambda)}{\partial \xi \partial \xi} = \lambda(1-\lambda)(\gamma_{H-1}^*)^3/(\sum_{j=1}^{H-1} \gamma_j^*) \times R$ by using Lemma 6 and the definition of $\xi$. $\square$

By setting $M = 1$ in LDA model, we have the sufficient conditions for mixture models.

**Corollary 4.** *For a mixture model, the same assertions as Theorem 3 hold for*

$$\tilde{R} = \sum_{n=1}^N \Omega_n'(f_{H-1}(X^{(n)} \mid \boldsymbol{\theta}_*^{(H-1)})) \frac{\partial^2 p(X^{(n)} \mid \eta_{H-1}^*)}{\partial \beta \partial \beta}. \tag{9}$$

*Proof.* For $M = 1$, $J_{\mu,\tau}^{(n)} = 0$ and $I^{(n)} = \gamma_{H-1}^* / \sum_{j=1}^{H-1} \gamma_j^*$. The assertion is obvious. $\square$

### 2.4   Critical Lines in Various Models

We further investigate the critical lines for specific models. Hereafter, we consider the maximum likelihood estimation, setting $\Omega_n(f) = \log f$ for all $n$.

**Gaussian Mixture, Mixture of Factor Analyzers, and Mixture of PCA**
Assume that each component is the $D$-dimensional Gaussian density with mean $\boldsymbol{\mu}$ and variance-covariance matrix $V$ as parameters, which is denoted by $\phi(\boldsymbol{x}; \boldsymbol{\mu}, V)$. The matrix $\tilde{R}$ in eq.(9) has a form $\tilde{R} = \begin{pmatrix} S_2 & S_3 \\ S_3^T & S_4 \end{pmatrix}$, where $S_2$, $S_3$, and $S_4$ correspond to the second derivatives with respect to $(\boldsymbol{\mu}, \boldsymbol{\mu})$, $(\boldsymbol{\mu}, V)$, and $(V, V)$, respectively. It is well known that the second derivative $\partial^2 \phi / \partial \boldsymbol{\mu} \partial \boldsymbol{\mu}$ of a Gaussian density is equal to the first derivative $\partial \phi / \partial V$. Then, $S_2$ is equal to zero by the condition of a critical point. If the data is randomly generated, $S_3$ and $S_4$ are of full rank almost surely. This type of matrix necessarily has a positive eigenvalue. It is not difficult to extend this discussion to models with scalar or diagonal variance-covariance matrices as variable parameters.

Similar arguments hold for mixture of factor analyzers (MFA, [4]) and mixture of probabilistic PCA (MPCA, [5]). In factor analyzers or probabilistic PCA, the variance-covariance matrix is restricted to the form

$$V = FF^T + S,$$

where $F$ is a factor loading of rank $k$ and $S$ is a diagonal or scalar matrix. Because the first derivative of $\phi(\boldsymbol{x}; \boldsymbol{\mu}, FF^T + S)$ with respect to $F$ is $\frac{\partial \phi(x; \mu, FF^T + S)}{\partial V} F$, the block in $\tilde{R}$ corresponding to the second derivatives on $\mu$ is not of full rank. In a similar manner to Gaussian mixtures, $\tilde{R}$ has a positive eigenvalue. In summary, we have the following

**Theorem 5.** *Suppose that a model is Gaussian mixture, MFA, or MPCA. If $\tilde{R}$ is of full rank, every point $\theta_\lambda$ on the critical line is a saddle point of $L_H$.*

This theorem means that if we have the maximum likelihood estimator for $H - 1$ components, we can find an ascending direction of likelihood by splitting a component and modifying their means and variance-covariance matrices in the direction of the positive eigenvector. This leads a component splitting method, which will be shown in Section 3.1.

**Latent Dirichlet Allocation**
We consider LDA with multinomial components. Using the $D$-dimensional random vector $x = (x_a) \in \{(1, 0, \ldots, 0)^T, \ldots, (0, \ldots, 0, 1)^T\}$, which indicates a chosen element, the multinomial distribution over $D$ elements is expressed as an exponential family by

$$p(x \,|\, \beta) = \prod_{a=1}^{D} (p_a)^{x_a} = \exp\{\textstyle\sum_{a=1}^{D-1} \beta^a x_a - \log(1 + \sum_{a=1}^{D-1} e^{\beta^a})\},$$

where $p_a$ is the expectation of $x_a$, and $\beta \in \mathbb{R}^{D-1}$ is a natural parameter given by $\beta^a = \log(p_a/p_D)$. It is easy to obtain

$$R = \textstyle\sum_{n=1}^{N} \Omega'(f_{H-1}(X^{(n)} \,|\, \boldsymbol{\theta}_*^{(H-1)})) \sum_{\mu=1}^{M} \sum_{\tau \neq \mu} J_{\mu,\tau}^{(n)} p(X_\mu^{(n)} \,|\, \gamma_{H-1}^*) p(X_\tau^{(n)} \,|\, \gamma_{H-1}^*)$$
$$\times (\tilde{X}_\mu^{(n)} - p_{(H-1)}^*)(\tilde{X}_\tau^{(n)} - p_{(H-1)}^*)^T, \quad (10)$$

where $\tilde{X}_\nu^{(n)}$ is the truncated $(D - 1)$-dimensional vector, and $p_{(H-1)}^* \in (0, 1)^{D-1}$ is the expectation parameter for $(H - 1)$-th component of $\boldsymbol{\theta}_*^{(H-1)}$.

In general, $J_{\mu,\tau}^{(n)}$ are intractable in large problems. We explain a simple case of $H = 2$ and $M = D$. Let $\widehat{p}$ be the frequency vector of the $D$ elements, which is the maximum likelihood estimator for the one multinomial model. In this case, we have $J_{\mu,\tau}^{(n)} = 1$ and

$$R = \textstyle\sum_{n=1}^{N} \{ \sum_{\mu,\tau=1}^{M} (\tilde{X}_\mu^{(n)} - \widehat{p})(\tilde{X}_\tau^{(n)} - \widehat{p})^T - \sum_{\mu=1}^{M} (\tilde{X}_\mu^{(n)} - \widehat{p})(\tilde{X}_\mu^{(n)} - \widehat{p})^T \}.$$

First, suppose we have a data set with $X_\nu^{(n)} = e_\nu$ for all $n$ and $1 \leq \nu \leq D = M$, where $e_j$ is the $D$-dimensional vector with the $j$-th component 1 and others zero. Then, we have $\widehat{p} = (1/D, \ldots, 1/D)$ and $\sum_{\mu=1}^{D} (\tilde{X}_\mu^{(n)} - \widehat{p}) = 0$, which means $R < 0$. The critical line gives maxima for LDA with $H = 2$. Next, suppose the data consists of $D$ groups, and every data in the $j$-th group is given by $X_\nu^{(n)} = e_j$. While we have again $\widehat{p} = (1/D, \ldots, 1/D)$, the matrix $R$ is $\sum_{j=1}^{D} (N/D) \times D(D - 1)(e_j - \widehat{p})(e_j - \widehat{p})^T > 0$. Thus, all the points on the critical lines are saddle points. These examples explain two extreme cases; in the former we have no advantage in using two components because all the data $X^{(n)}$ are the same, while in the latter the multiple components fits better to the variety of $X^{(n)}$.

## 3 Component Splitting Method in Mixture of Gaussian Components

### 3.1 EM with Component Splitting

It is well known that the EM algorithm suffers from strong dependency on initialization. In addition, because the likelihood of a mixture of Gaussian components is not upper bounded

---

**Algorithm 1 : EM with component splitting for Gaussian mixture**

---

1. Initialization: calculate the sample mean $\mu_1$ and variance-covariance matrix $V_1$.

2. $H := 1$.

3. For all $1 \leq h \leq H$, diagonalize $V_{h*}$ as $V_{h*} = U_h \Lambda_h U_h^T$, and calculate $\tilde{R}_h$ according to eq.(12) in Appendix.

4. For $1 \leq h \leq H$, calculate the eigenvector $(r_h, W_h)$ of $\tilde{R}_h$ corresponding to the largest eigenvalue.

5. For $1 \leq h \leq H$, optimize $\beta$ by line search to maximize the likelihood for

$$
\begin{aligned}
c_h &= \tfrac{1}{2} c_{h*}, & \mu_h &= \mu_{h*} - \beta r_h, & V_h &= U_h e^{-\beta W_h} \Lambda_h e^{-\beta W_h} U_h^T, \\
c_{H+1} &= \tfrac{1}{2} c_{h*}, & \mu_{H+1} &= \mu_{h*} + \beta r_h, & V_{H+1} &= U_h e^{\beta W_h} \Lambda_h e^{\beta W_h} U_h^T.
\end{aligned}
\tag{11}
$$

Let $\beta_h^o$ be the optimizer and $L_h$ be the likelihood.

6. For $h^\dagger := \arg\max_h L_h$, split $h^\dagger$-th component according to eq.(11) with $\beta_{h^\dagger}^o$.

7. Optimize the parameter $\boldsymbol{\theta}^{(H+1)}$ using EM algorithm. Let $\boldsymbol{\theta}_*^{(H+1)}$ be the result.

8. If $H + 1 = \text{MAX\_H}$, then END. Otherwise, $H := H + 1$ and go to 3.

---

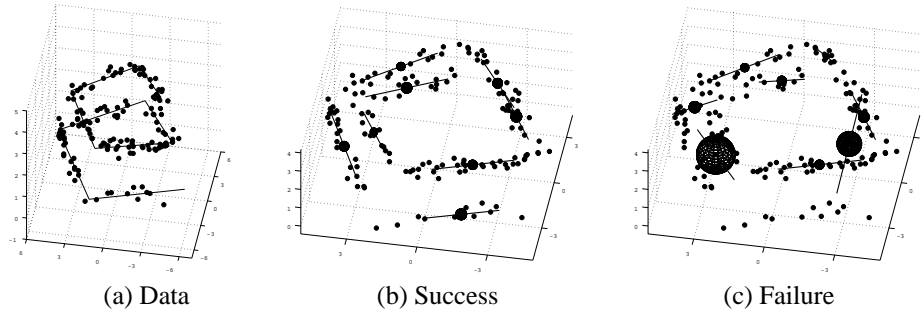

| (a) Data | (b) Success | (c) Failure |

Figure 1: Spiral data. In (b) and (c), the lines represent the factor loading vectors $F_h$ and $-F_h$ at the mean values, and the radius of a sphere is the scalar part of the variance.

for small variances, we should use an optimization technique to give an appropriate maximum. Sequential split of components can give a solution to these problems. From Theorem 5, a stable and effective way of splitting a Gaussian component is derived to increase the likelihood. We propose EM with component splitting, which adds a component one by one after maximizing the likelihood at each size. Ueda et al ([6]) proposes Split and Merge EM, in which the components repeat split and merge in a triplet, keeping the total number fixed. While their method works well, it requires a large number of trials of EM for candidate triplets, and the splitting method is heuristic. Our splitting method is well based on theory, and EM with splitting gives a series of estimators for all model sizes in a single run.

Algorithm 1 is the procedure of learning. We show only the case of mixture of Gaussian. The exact algorithm for the mixture of PCA/FA will be shown in a forthcoming paper. It is noteworthy that in splitting a component, not only the means but also the variance-covariance matrices must be modified. The simple additive rule $V_{new} = V_{old} + \Delta V$ tends to fail, because it may make the matrix non-positive definite. To solve this problem, we use Lie algebra expression to add a vector of ascending direction. Let $V = U \Lambda U^T$ be the diagonalization of $V$, and consider $V(W) = U e^W \Lambda e^W U^T$ for a symmetric matrix

$W$. This gives a local coordinate of the positive definite matrices around $V = V(0)$. Modification of $V$ through $W$ gives a stable way of updating variance-covariance matrices.

## 3.2 Experimental results

We show through experiments how the proposed EM with component splitting effectively maximizing the likelihood. In the first experiment, the mixture of PCA with 8 components of rank 1 is employed to fit the synthesized 150 data generated along a piecewise linear spiral (Fig.1). Table 1-(a) shows the results over 30 trials with different random numbers. We use the on-line EM algorithm ([7]), presenting data one-by-one in a random order. The EM with random initialization reaches the best state (Fig.1-(b)) only 6 times, while EM with component splitting achieves it 26 times. Fig.1-(c) shows an example of failure.

The next experiment is an image compression problem, in which the image "Lenna" of $160{\times}160$ pixels (Fig.2) is used. The image is partitioned into $20{\times}20$ blocks of $8{\times}8$ pixels, which are regarded as 400 data in $\mathbb{R}^{64}$. We use the mixture of PCA with 10 components of rank 4, and obtain a compressed image by $\hat{X} = F_h(F_h^T F_h)^{-1} F_h^T X$, where $X$ is a 64 dimensional block and $h$ indicates the component of the shortest Euclidean distance $\|X - \mu_h\|$. Table 1-(b) shows the residual square error (RSE), $\sum_{j=1}^{400} \|X_j - \hat{X}_j\|^2$, which shows the quality of the compression. In both experiments, we can see the better optimization performance of the proposed algorithm.

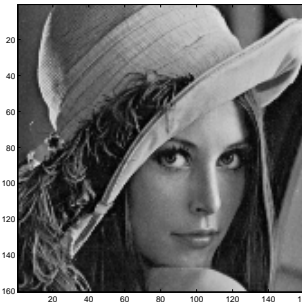

Figure 2: "Lenna".

|  | (a) Likelihood for spiral data (30 runs) | | | (b) RSE for "Lenna" (10 runs) | |
|---|---|---|---|---|---|
|  | EM | EMCS | $\times 10^4$ | EM | EMCS |
| Best | -534.9 (6 times) | -534.9 (26 times) | Best | 5.94 | 5.38 |
| Worst | -648.1 | -587.9 | Worst | 6.40 | 6.12 |
| Av. | -583.9 | -541.3 | Av. | 6.15 | 5.78 |

Table 1: Experimental results. EM is the conventional EM with random initialization, and EMCS is the proposed EM with component splitting.

# 4 Discussions

In EM with component splitting, we obtain the estimators up to the specified number of components. We need a model selection technique to choose the best one, which is another important problem. We do not discuss it in this paper, because our method can be combined with many techniques, which select a model after obtaining the estimators. However, we should note that some famous methods such as AIC and MDL, which are based on statistical asymptotic theory, cannot be applied to mixture models because of the unidentifiability of the parameter. Further studies are necessary on model selection for mixture models.

Although the computation to calculate the matrix $R$ is not cheap in a mixture of Gaussian components, the full variance-covariance matrices are not always necessary in practical problems. It can save the computation drastically. Also, some methods to reduce the computational cost should be more investigated.

In selecting a component to split, we try line search for all the components and choose the one giving the largest likelihood. While this works well in our experiments, the proposed method of component splitting can be combined with other criterions to select a component.

One of them is to select the component giving the largest eigenvalue of $\tilde{R}_h$. In Gaussian mixture models, this is very natural; the block of the second derivatives w.r.t. $V$ in $\tilde{R}$ is equal to the weighted fourth cummulant, and a component with a large cummulant should be split. However, in mixture of FA and PCA, this does not necessarily work well, because the decomposition $V = FF^T + S$ does not give a natural parametrization. Although we have discussed only local properties, a method incorporating global information might be more preferable. These are left as a future work.

## Appendix

**Lemma 6.** *Suppose* $\varphi_H(\boldsymbol{u}^{(H)}; \boldsymbol{\beta}^{(H)})$ *satisfies the assumption (S-1). Define* $I_H(\boldsymbol{\alpha}^{(H)}; \boldsymbol{\beta}^{(H)}) = \int_{\Delta_{H-1}} \varphi(\boldsymbol{u}^{(H)}; \boldsymbol{\beta}^{(H)}) \mathcal{D}_H(\boldsymbol{u}^{(H)} \mid \boldsymbol{\alpha}^{(H)}) d\boldsymbol{u}^{(H)}$. *Then,* $I_H$ *also satisfies (S-1);*

$$I_H(\boldsymbol{\alpha}^{(H)}; \boldsymbol{\beta}^{(H-2)}, \beta_{H-1}, \beta_{H-1}) = I_{H-1}(\boldsymbol{\alpha}^{(H-2)}, \alpha_{H-1} + \alpha_H; \boldsymbol{\beta}^{(H-1)}).$$

*Proof.* Direct calculation. $\qquad\square$

### Matrix $\tilde{R}_h$ for Gaussian mixture

We omit the index $h$ for simplicity, and use Einstein's convention. Let $U = (u_1, \ldots, u_D)$ and $\Lambda = \mathrm{diag}(\lambda_1, \ldots, \lambda_D)$. For $V(W) = Ue^W \Lambda e^W U^T$, we have $\partial V(O)/\partial W_{ab} = (\lambda_a + (1 - \delta_{ab})\lambda_b)(u_a u_b^T + u_b u_a^T)$, where $\delta_{ab}$ is Kronecker's delta. Let $T^{(3)}$ and $T^{(4)}$ be the weighted third and fourth sample moments, respectively, with weights $\frac{\phi(x^{(n)}; \mu_*, V_*)}{f^{(H-1)}(x^{(n)}; \boldsymbol{\theta}_*^{(H-1)})}$. $\tilde{T}_{(3)}$ and $\tilde{T}_{(4)}$ are defined by $\tilde{T}_{(3)}^{abc} = V^{ap} V^{bq} V^{cr} T_{pqr}^{(3)}$ and $\tilde{T}_{(4)}^{abcd} = V^{ap} V^{bq} V^{cr} V^{ds} T_{pqrs}^{(4)}$, respectively, where $V^{ap}$ is the $(ap)$-component of $V^{-1}$. Direct calculation leads that the matrix $\tilde{R} = \begin{pmatrix} O & B \\ B^T & C \end{pmatrix}$, where the decomposition corresponds to $\beta = (\mu, W)$, is given by

$$B_{\mu_a, W_{bc}} = (\lambda_b + (1 - \delta_{bc})\lambda_c) u_b^T \tilde{T}_{(3)}^{\cdot\cdot a} u_c$$

$$C_{W_{ab} W_{cd}} = (\lambda_a u_b u_a^T + (1 - \delta_{ab})\lambda_b u_a u_b^T)_{pq} (\lambda_c u_d u_c^T + (1 - \delta_{cd})\lambda_d u_c u_d^T)_{rs} \quad (12)$$
$$\times \{ \tilde{T}_{(4)}^{pqrs} - (V^{pq} V^{rs} + V^{pr} V^{qs} + V^{ps} V^{qr}) \}.$$

In the above equation, $\tilde{T}_{(3)}^{\cdot\cdot a}$ is the $D \times D$ matrix with fixed $a$ for $\tilde{T}_{(3)}^{bca}$.

## Footnotes

[1]The results do not require that $p(x \mid \beta)$ is a density function. Thus, they can be easily extended to function fitting in regression, which gives the results on multilayer neural networks in [1].

## References

[1] K. Fukumizu and S. Amari. Local minima and plateaus in hierarchical structures of multilayer perceptrons. *Neural Networks*, 13(3):317–327, 2000.

[2] D. M. Blei, A. Y. Ng, and M. I. Jordan. Latent Dirichlet allocation. *Advances in Neural Information Processing Systems*, 14, 2002. MIT Press.

[3] S. Amari, H. Park, and T. Ozeki. Geometrical singularities in the neuromanifold of multilayer perceptrons. *Advances in Neural Information Processing Systems*, 14, 2002. MIT Press.

[4] Z. Ghahramani and G. Hinton. The EM algorithm for mixtures of factor analyzers. Technical Report CRG-TR-96-1, University of Toronto, Department of Computer Science, 1997.

[5] M. Tipping and C. Bishop. Mixtures of probabilistic principal component analysers. *Neural Computation*, 11:443–482, 1999.

[6] N. Ueda, R. Nakano, Z. Ghahramani, and G. Hinton. SMEM algorithm for mixture models. *Neural Computation*, 12(9):2109–2128, 2000.

[7] M. Sato and S. Ishii. On-line EM algorithm for the normalized Gaussian network. *Neural Computation*, 12(2):2209–2225, 2000.
